# Spike Frequency Adaptation Implements Anticipative Tracking in Continuous Attractor Neural Networks

**Yuanyuan Mi**
State Key Laboratory of Cognitive Neuroscience & Learning,
Beijing Normal University,Beijing 100875,China
miyuanyuan0102@bnu.edu.cn

**C. C. Alan Fung,    K. Y. Michael Wong**
Department of Physics, The Hong Kong University of Science and Technology, Hong Kong
phccfung@ust.hk,   phkywong@ust.hk

**Si Wu**
State Key Laboratory of Cognitive Neuroscience & Learning, IDG/McGovern Institute for
Brain Research, Beijing Normal University, Beijing 100875, China
wusi@bnu.edu.cn

## Abstract

To extract motion information, the brain needs to compensate for time delays that are ubiquitous in neural signal transmission and processing. Here we propose a simple yet effective mechanism to implement anticipative tracking in neural systems. The proposed mechanism utilizes the property of spike-frequency adaptation (SFA), a feature widely observed in neuronal responses. We employ continuous attractor neural networks (CANNs) as the model to describe the tracking behaviors in neural systems. Incorporating SFA, a CANN exhibits intrinsic mobility, manifested by the ability of the CANN to support self-sustained travelling waves. In tracking a moving stimulus, the interplay between the external drive and the intrinsic mobility of the network determines the tracking performance. Interestingly, we find that the regime of anticipation effectively coincides with the regime where the intrinsic speed of the travelling wave exceeds that of the external drive. Depending on the SFA amplitudes, the network can achieve either perfect tracking, with zero-lag to the input, or perfect anticipative tracking, with a constant leading time to the input. Our model successfully reproduces experimentally observed anticipative tracking behaviors, and sheds light on our understanding of how the brain processes motion information in a timely manner.

## 1   Introduction

Over the past decades, our knowledge of how neural systems process static information has advanced considerably, as is well documented by the receptive field properties of neurons. The equally important issue of how neural systems process motion information remains much less understood. A main challenge in processing motion information is to compensate for time delays that are pervasive in neural systems. For instance, visual signal transmitting from the retina to the primary visual cortex takes about 50-80 ms [1], and the time constant for single neurons responding to synaptic input is of the order 10-20 ms [2]. If these delays are not compensated properly, our perception of a fast moving object will lag behind its true position in the external world significantly, impairing our vision and motor control.

A straightforward way to compensate for time delays is to anticipate the future position of a moving object, covering the distance the object will travel through during the delay period. Experimental data has suggested that our brain does employ such a strategy. For instance, it was found that in spatial navigation, the internal head-direction encoded by anterior dorsal thalamic nuclei (ADN) cells in a rodent was leading the instant position of the rodent's head by $\sim 25$ ms [3, 4, 5], i.e., it was the direction the rodent's head would turn into $\sim 25$ ms later. Anticipation also justifies the well-known flash-lag phenomenon [6], that is, the perception that a moving object leads a flash, although they coincide with each other at the same physical location. The reason is due to the anticipation of our brain for the future position of the continuously moving object, in contrast to the lack of anticipation for intermittent flashes. Although it is clear that the brain do have anticipative response to the animal's head direction, it remains unclear how neural systems implement appropriate anticipations against various forms of delays.

Depending on the available information, the brain may employ different strategies to implement anticipations. In the case of self-generated motion, the brain may use an efference copy of the motor command responsible for the motion to predict the motion consequence in advance [7]; and in the case when there is an external visual cue, such as the speed of a moving object, the neural system may dynamically select a transmission route which sends the object information directly to the future cortical location during the delay [8]. These two strategies work well in their own feasible conditions, but they may not compensate for all kinds of neural delays, especially when the internal motor command and visual cues are not available. Notably, it was found that when a rodent was moving passively, i.e., a situation where no internal motor command is available, the head-direction encoded by ADN cells was still leading the actual position of the rodent's head by around $\sim 50$ms, even larger than that in a free-moving condition [5]. Thus, extra anticipation strategies may exist in neural systems.

Here, we propose a novel mechanism to generate anticipative responses when a neural system is tracking a moving stimulus. This strategy does not depend on the motor command information nor external visual cues, but rather relies on the intrinsic property of neurons, i.e., spike-frequency adaptation (SFA). SFA is a dynamical feature commonly observed in the activities of neurons when they have experienced prolonged firing. It may be generated by a number of mechanisms [10]. In one mechanism, neural firing elevates the intracellular calcium level of a neuron, which induces an inward potassium current and subsequently hyperpolarizes the neuronal membrane potential [11]. In other words, strong neuronal response induces a negative feedback to counterbalance itself. In the present study, we use continuous attractor neural networks (CANNs) to model the tracking behaviors in neural systems. It was known that SFA can give rise to travelling waves in CANNs [12] analogous to the effects of asymmetric neuronal interactions; here we will show that its interplay with external moving stimuli determines the tracking performance of the network. Interestingly, we find that when the intrinsic speed of the network is greater than that of the external drive, anticipative tracking occurs for sufficiently weak stimuli; and different SFA amplitude results in different anticipative times.

## 2   The Model

### 2.1   Continuous attractor neural networks

We employ CANNs as the model to investigate the tracking behaviors in neural systems. CANNs have been successfully applied to describe the encoding of continuous stimuli in neural systems, including orientation [13], head-direction [14], moving direction [15] and self location [16]. Recent experimental data strongly indicated that CANNs capture some fundamental features of neural information representation [17].

Consider a one-dimensional continuous stimulus $x$ encoded by an ensemble of neurons (Fig. 1). The value of $x$ is in the range of $(-\pi, \pi]$ with the periodic condition imposed. Denote $U(x, t)$ as the synaptic input at time $t$ of the neurons whose preferred stimulus is $x$, and $r(x, t)$ the corresponding firing rate. The dynamics of $U(x, t)$ is determined by the recurrent input from other neurons, its own relaxation and the external input $I^{\text{ext}}(x, t)$, which is written as

$$\tau \frac{dU(x, t)}{dt} = -U(x, t) + \rho \int_{x'} J(x, x')r(x', t)dx' + I^{\text{ext}}(x, t), \qquad (1)$$

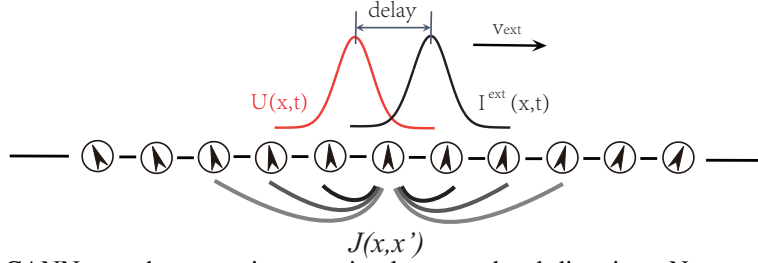

Figure 1: A CANN encodes a continuous stimulus, e.g., head-direction. Neurons are aligned in the network according to their preferred stimuli. The neuronal interaction $J(x, x')$ is translation-invariant in the space of stimulus values. The network is able to track a moving stimulus, but the response bump $U(x, t)$ is always lagging behind the external input $I^{\text{ext}}(x, t)$ due to the neural response delay.

where $\tau$ is the synaptic time constant, typically of the order $2 \sim 5$ ms, $\rho$ is the neural density and $J(x, x') = \frac{J_0}{\sqrt{2\pi}a} \exp\left[-(x - x')^2/(2a^2)\right]$ is the neural interaction from $x'$ to $x$, where the Gaussian width $a$ controls the neuronal interaction range. We will consider $a \ll \pi$. Under this condition, the neuronal responses are localized and we can effectively treat $-\infty < x < \infty$ in the following analysis.

The nonlinear relationship between $r(x, t)$ and $U(x, t)$ is given by

$$r(x, t) = \frac{U(x, t)^2}{1 + k\rho \int_{x'} U(x', t)^2 dx'}, \tag{2}$$

where the divisive normalization could be realized by shunting inhibition. $r(x, t)$ first increases with $U(x, t)$ and then saturates gradually when the total network activity is sufficiently large. The parameter $k$ controls the strength of divisive normalization. This choice of global normalization can simplify our analysis and should not alter our main conclusion if localized inhibition is considered.

It can be checked that when $I^{\text{ext}} = 0$, the network supports a continuous family of Gaussian-shaped stationary states, called bumps, which are,

$$\overline{U}(x) = U_0 \exp\left[-\frac{(x - z)^2}{4a^2}\right], \quad \overline{r}(x) = r_0 \exp\left[-\frac{(x - z)^2}{2a^2}\right], \quad \forall z \tag{3}$$

where the peak position of the bump $z$ is a free parameter. $r_0 = \sqrt{2}U_0/(\rho J_0)$ and $U_0 = [1 + \sqrt{1 - 8\sqrt{2\pi}ak/(\rho J_0^2)}]/(2\sqrt{2\pi}ak\rho)$. The bumps are stable for $0 < k < k_c$, with $k_c = \rho J_0^2/(8\sqrt{2\pi}a)$.

The bump states of a CANN form a sub-manifold in the state space of the network, on which the network is neutrally stable. This property enables a CANN to track a moving stimulus smoothly, provided that the stimulus speed is not too large [18]. However, during the tracking, the network bump is always lagging behind the instant position of the moving stimulus due to the delay in neuronal responses (Fig. 1).

## 2.2 CANNs with the asymmetrical neuronal interaction

It is instructive to look at the dynamical properties of a CANN when the asymmetrical neuronal interaction is included. In an influential study [14], Zhang proposed an idea of adding asymmetrical interactions between neurons in a CANN, such that the network can support travelling waves, i.e., spontaneously moving bumps. The modified model well describes the experimental finding that in tracking the rotation of a rodent, the internal representation of head-direction constructed by ADN cells also rotates and the bump of neural population activity remains largely invariant in the rotating frame.

By including the asymmetrical neuronal interaction, the CANN model presented above also supports a travelling wave state. The new neuronal recurrent interaction is written as

$$\tilde{J}(x, x') = \frac{J_0}{\sqrt{2\pi}a} \exp\left[-\frac{(x - x')^2}{2a^2}\right] + \gamma\tau \frac{J_0}{\sqrt{2\pi}a^3}(x - x') \exp\left[-\frac{(x - x')^2}{2a^2}\right], \tag{4}$$

where $\gamma$ is a constant controlling the strength of asymmetrical interaction.

It is straightforward to check that the network supports the following traveling wave solution, $\overline{U}(x,t) = U_0 \exp\left\{-[x-(z+vt)]^2/(4a^2)\right\}, \quad \overline{r}(x,t) = r_0 \exp\left\{-[x-(z+vt)]^2/(2a^2)\right\}$, where $v$ is the speed of the travelling wave, and $v = \gamma$, i.e., the asymmetrical interaction strength determines the speed of the travelling wave (see Supplementary Information).

### 2.3 CANNs with SFA

The aim of the present study is to explore the effect of SFA on the tracking behaviors of a CANN. Incorporating SFA, the dynamics of a CANN is written as

$$\tau \frac{dU(x,t)}{dt} = -U(x,t) + \rho \int_{x'} J(x,x')r(x',t)dx' - V(x,t) + I^{\text{ext}}(x,t), \tag{5}$$

where the synaptic current $V(x,t)$ represents the effect of SFA, whose dynamics is given by [12]

$$\tau_v \frac{dV(x,t)}{dt} = -V(x,t) + mU(x,t), \tag{6}$$

where $\tau_v$ is the time constant of SFA, typically of the order $40 \sim 120$ ms. The parameter $m$ controls the SFA amplitude. Eq. (6) gives rise to $V(x,t) = m \int_{-\infty}^{t} \exp\left[-(t-t')/\tau_v\right] U(x,t')dt'/\tau_v$, that is, $V(x,t)$ is the integration of the neuronal synaptic input (and hence the neuronal activity) over an effective period of $\tau_v$. The negative value of $V(x,t)$ is subsequently fed back to the neuron to suppress its response (Fig. 2A). The higher the neuronal activity level is, the larger the negative feedback will be. The time constant $\tau_v \gg \tau$ indicates that SFA is slow compared to neural firing.

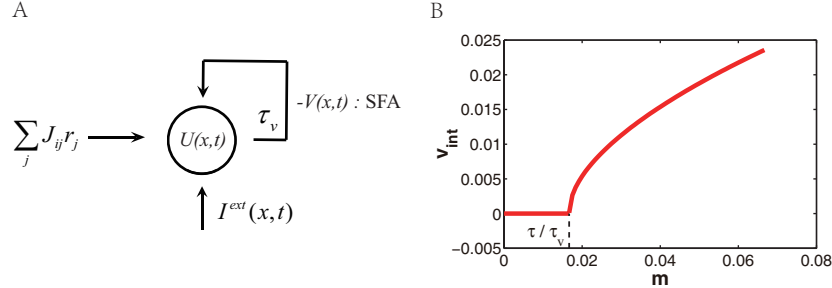

Figure 2: A. The inputs to a single neuron in a CANN with SFA, which include a recurrent input $\sum_j J_{ij}r_j$ from other neurons, an external input $I^{\text{ext}}(x,t)$ containing the stimulus information, and a negative feedback current $-V(x,t)$ representing the SFA effect. The feedback of SFA is effectively delayed by time $\tau_v$. B. The intrinsic speed of the network $v_{\text{int}}$ (in units of $1/\tau$) increases with the SFA amplitude $m$. The network starts to have a travelling wave state at $m = \tau/\tau_v$. The parameters are: $\tau = 1, \tau_v = 60, a = 0.5$. Obtained by Eq. (13).

## 3 Travelling Wave in a CANN with SFA

We find that SFA has the same effect as the asymmetrical neuronal interaction on retaining travelling waves in a CANN. The underlying mechanism is intuitively understandable. Suppose that a bump emerges at an arbitrary position in the network. Due to SFA, those neurons which are most active receive the strongest negative feedback, and their activities will be suppressed accordingly. Under the competition (mediated by recurrent connections and divisive normalization) from the neighboring neurons which are less affected by SFA, the bump tends to shift to the neighborhood; and at the new location, SFA starts to destabilize neuronal responses again. Consequently, the bump will keep moving in the network like a travelling wave.

The condition for the network to support a travelling wave state can be theoretically analyzed. In simulations, we observe that in a traveling wave state, the profiles of $U(x,t)$ and $V(x,t)$ have approximately a Gaussian shape, if $m$ is small enough. We therefore consider the following Gaussian

ansatz for the travelling wave state,

$$\overline{U}(x,t) = A_u \exp\left\{-\frac{[x-z(t)]^2}{4a^2}\right\}, \tag{7}$$

$$\overline{r}(x,t) = A_r \exp\left\{-\frac{[x-z(t)]^2}{2a^2}\right\}, \tag{8}$$

$$\overline{V}(x,t) = A_v \exp\left\{-\frac{[x-(z(t)-d)]^2}{4a^2}\right\}, \tag{9}$$

where $dz(t)/dt$ is the speed of the travelling wave and $d$ is the separation between $\overline{U}(x,t)$ and $\overline{V}(x,t)$. Without loss of generality, we assume that the bump moves from left to right, i.e., $dz(t)/dt > 0$. Since $V(x,t)$ lags behind $U(x,t)$ due to slow SFA, $d > 0$ normally holds.

To solve the network dynamics, we utilize an important property of CANNs, that is, the dynamics of a CANN are dominated by a few motion modes corresponding to different distortions in the shape of a bump [18]. We can project the network dynamics onto these dominating modes and simplify the network dynamics significantly. The first two dominating motion modes used in the present study correspond to the distortions in the height and position of the Gaussian bump, which are given by $\phi_0(x|z) = \exp\left[-(x-z)^2/(4a^2)\right]$ and $\phi_1(x|z) = (x-z)\exp\left[-(x-z)^2/(4a^2)\right]$. By projecting a function $f(x)$ onto a mode $\phi_n(x)$, we mean computing $\int_x f(x)\phi_n(x)dx/\int_x \phi_n(x)^2 dx$.

Applying the projection method, we solve the network dynamics and obtain the travelling wave state. The speed of the travelling wave and the bumps' separation are calculated to be (see Supplementary Information)

$$d = 2a\sqrt{1-\sqrt{\frac{\tau}{m\tau_v}}}, \quad v_{\text{int}} \equiv \frac{dz(t)}{dt} = \frac{2a}{\tau_v}\sqrt{\frac{m\tau_v}{\tau} - \sqrt{\frac{m\tau_v}{\tau}}}. \tag{10}$$

The speed of the travelling wave reflects the intrinsic mobility of the network, and its value is fully determined by the network parameters (see Eq. (10)). Hereafter, we call it the intrinsic speed of the network, referred to as $v_{\text{int}}$. $v_{\text{int}}$ increases with the SFA amplitude $m$ (Fig. 2B). The larger the value of $v_{\text{int}}$, the higher the mobility of the network.

From the above equations, we see that the condition for the network to support a travelling wave state is $m > \tau/\tau_v$. We note that SFA effects can reduce the firing rate of neurons significantly [11]. Since the ratio $\tau/\tau_v$ is small, it is expected that this condition can be realistically fulfilled.

### 3.1 Analogy to the asymmetrical neuronal interaction

Both SFA and the asymmetrical neuronal interaction have the same capacity of generating a travelling wave in a CANN. We compare their dynamics to unveil the underlying cause.

Consider that the network state is given by Eq. (8). The contribution of the asymmetrical neuronal interaction can be written as (substituting the asymmetrical component in Eq. (4) into the second term on the right-hand side of Eq. (1)),

$$\frac{J_0\rho\gamma\tau r_0}{\sqrt{2\pi}a^3}\int_{x'}(x-x')e^{-\frac{(x-x')^2}{2a^2}}e^{-\frac{(x'-z)^2}{2a^2}}dx' = \frac{\rho J_0 r_0\gamma\tau(x-z)}{2\sqrt{2}a^2}e^{-\frac{(x-z)^2}{4a^2}}. \tag{11}$$

In a CANN with SFA, when the separation $d$ is sufficiently small, the synaptical current induced by SFA can be approximately expressed as (the 1st-order Taylor expansion; see Eq. (9)),

$$-\overline{V}(x,t) \approx -A_v \exp\left[-\frac{(x-z)^2}{4a^2}\right] + dA_v\frac{x-z}{2a^2}\exp\left[-\frac{(x-z)^2}{4a^2}\right], \tag{12}$$

which consists of two terms: the first one has the same form as $\overline{U}(x,t)$ and the second one has the same form as the contribution of the asymmetrical interaction (compared to Eq. (11)). Thus, SFA has the similar effect as the asymmetrical neuronal interaction on the network dynamics.

The notion of the asymmetrical neuronal interaction is appealing for retaining a travelling wave in a CANN, but its biological basis has not been properly justified. Here, we show that SFA may provide

an effective way to realize the effect of the asymmetrical neuronal interaction without recruiting the hard-wired asymmetrical synapses between neurons. Furthermore, SFA can implement travelling waves in either direction, whereas, the hard-wired asymmetrical neuronal connections can only support a travelling wave in one direction along the orientation of the asymmetry. Consequently, a CANN with the asymmetric coupling can only anticipatively track moving objects in one direction.

## 4  Tracking Behaviors of a CANN with SFA

SFA induces intrinsic mobility of the bump states of a CANN, manifested by the ability of the network to support self-sustained travelling waves. When the network receives an external input from a moving stimulus, the tracking behavior of the network will be determined by two competing factors: the intrinsic speed of the network ($v_{\text{int}}$) and the speed of the external drive ($v_{\text{ext}}$). Interestingly, we find that when $v_{\text{int}} > v_{\text{ext}}$, the network bump leads the instant position of the moving stimulus for sufficiently weak stimuli, achieving anticipative tracking.

Without loss of generality, we set the external input to be $I^{\text{ext}}(x,t) = \alpha \exp\left\{-\left[x - z_0(t)\right]^2/(4a^2)\right\}$, where $\alpha$ represents the input strength, $z_0(t)$ is the stimulus at time $t$ and the speed of the moving stimulus is $v_{\text{ext}} = dz_0(t)/dt$.

Define $s = z(t) - z_0(t)$ to be the displacement of the network bump relative to the external drive. We consider that the network is able to track the moving stimulus, i.e., the network dynamics will reach a stationary state with $dz(t)/dt = dz_0(t)/dt$ and $s$ a constant. Since we consider that the stimulus moves from left to right, $s > 0$ means that the network tracking is leading the moving input; whereas $s < 0$ means the network tracking is lagging behind.

Using the Gaussian ansatz for the network state as given by Eqs. (7-9) and applying the projection method, we solve the network dynamics and obtain (see Supplementary Information),

$$d = 2a\frac{-a + \sqrt{a^2 + (v_{\text{ext}}\tau_v)^2}}{v_{\text{ext}}\tau_v}, \tag{13}$$

$$s \exp\left(-\frac{s^2}{8a^2}\right) = \frac{1}{\alpha}A_u\frac{\tau}{v_{\text{ext}}}\left(\frac{md^2}{\tau\tau_v} - v_{\text{ext}}^2\right). \tag{14}$$

Combining Eqs. (10, 13, 14), it can be checked that when $v_{\text{ext}} = v_{\text{int}}$, $v_{\text{ext}}^2 = md^2/(\tau\tau_v)$, which gives $s = 0$ ; and when $v_{\text{ext}} < v_{\text{int}}$, $v_{\text{ext}}^2 < md^2/(\tau\tau_v)$, which gives $s > 0$, i.e., the bump is leading the external drive (For detail, see Supplementary Information).

Fig. 3A presents the simulation result. There is a minor discrepancy between the theoretical prediction and the simulation result: the separation $s = 0$ happens at the point when the stimulus speed $v_{\text{ext}}$ is slightly smaller than the intrinsic speed of the network $v_{\text{int}}$. This discrepancy arises from the distortion of the bump shape from Gaussian when the input strength is strong, the stimulus speed is high and $m$ is large, and hence the Gaussian ansatz on the network state is not accurate. Nevertheless, for sufficiently weak stimuli, the theoretical prediction is correct.

### 4.1  Perfect tracking and perfect anticipative tracking

As observed in experiments, neural systems can compensate for time delays in two different ways: 1) perfect tracking, in which the network bump has zero-lag with respect to the external drive, i.e., $s = 0$; and 2) perfect anticipative tracking, in which the network bump leads the external drive by approximately a constant time $t_{\text{ant}} = s/v_{\text{ext}}$. In both cases, the tracking performance of the neural system is largely independent of the stimulus speed. We check whether a CANN with SFA exhibits these appealing properties.

Define a scaled speed variable $\overline{v}_{\text{ext}} \equiv \tau_v v_{\text{ext}}/a$. In a normal situation, $\overline{v}_{\text{ext}} \ll 1$. For instance, taking the biologically plausible parameters $\tau_v = 100$ ms and $a = 50^{\text{o}}$, $\overline{v}_{\text{ext}} = 0.1$ corresponds to $v_{\text{ext}} = 500^{\text{o}}/s$, which is a rather high speed for a rodent rotating its head in ordinary life.

By using the scaled speed variable, Eq. (14) becomes

$$s \exp\left(-\frac{s^2}{8a^2}\right) = \frac{1}{\alpha}A_u a\left[4m\frac{(-1 + \sqrt{1 + \overline{v}_{\text{ext}}^2})^2}{\overline{v}_{\text{ext}}^3} - \frac{\tau}{\tau_v}\overline{v}_{\text{ext}}\right]. \tag{15}$$

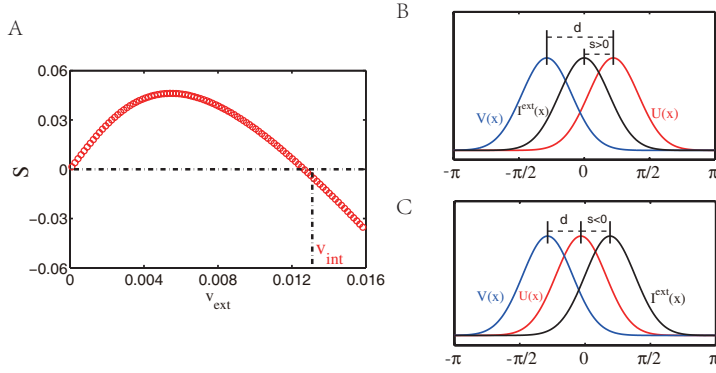

Figure 3: A. The separation $s$ vs. the speed of the external input $v_{\text{ext}}$. Anticipative tracking $s > 0$ occurs when $v_{\text{ext}} < v_{\text{int}}$. The simulation was done with a network of $N = 1000$ neurons. The parameters are: $J_0 = 1, k = 0.1, a = 0.5, \tau = 1, \tau_v = 60$, $\alpha = 0.5$ and $m = 2.5\tau/\tau_v$. B. An example of anticipative tracking in the reference frame of the external drive. C. An example of delayed tracking. In both cases, the profile of $V(x,t)$ is lagging behind the bump $U(x,t)$ due to slow SFA.

In the limit of $\overline{v}_{\text{ext}} \ll 1$ and consider $s/(2\sqrt{2}a) \ll 1$ (which is true in practice), we get $s \approx A_u \tau_v v_{\text{ext}}(m - \frac{\tau}{\tau_v})/\alpha$. Thus, we have the following two observations:

- **Perfect tracking**. When $m \approx \tau/\tau_v$, $s \approx 0$ holds, and perfect tracking is effectively achieved. Notably, when there is no stimulus, $m = \tau/\tau_v$ is the condition for the network starting to have a traveling wave state.

- **Perfect anticipative tracking**. When $m > \tau/\tau_v$, $s$ increases linearly with $v_{\text{ext}}$, and the anticipative time $t_{\text{ant}}$ is approximately a constant.

These two properties hold for a wide range of stimulus speed, as long as the approximation $\overline{v}_{\text{ext}} \ll 1$ is applicable. We carried out simulations to confirm the theoretical analysis, and the results are presented in Fig. 4. We see that: (1) when SFA is weak, i.e., $m < \tau/\tau_v$, the network tracking is lagging behind the external drive, i.e. $s < 0$ (Fig. 4A); (2) when the amplitude of SFA increases to a critical value $m = \tau/\tau_v$, $s$ becomes effectively zero for a wide range of stimulus speed, and perfect tracking is achieved (Fig. 4B); (3) when SFA is large enough satisfying $m > \tau/\tau_v$, $s$ increases linearly with $v_{\text{ext}}$ for a wide range of stimulus speeds, achieving perfect anticipative tracking (Fig. 4C); and (4) with the increasing amplitude of SFA, the anticipative time of the network also increases (Fig. 4D). Notably, by choosing the parameters properly, our model can replicate the experimental finding on a constant leading time of around 25 ms when a rodent was tracking head-direction by ADN cells (the red points in Fig. 4D for $\tau = 5$ ms) [19].

## 5 Conclusions

In the present study, we have proposed a simple yet effective mechanism to implement anticipative tracking in neural systems. The proposed strategy utilizes the property of SFA, a general feature in neuronal responses, whose contribution is to destabilize spatially localized attractor states in a network. Analogous to asymmetrical neuronal interactions, SFA induces self-sustained travelling wave in a CANN. Compared to the former, SFA has the advantage of not requiring the hard-wired asymmetrical synapses between neurons. We systematically explored how the intrinsic mobility of a CANN induced by SFA affects the network tracking performances, and found that: (1) when the intrinsic speed of the network (i.e., the speed of the travelling wave the network can support) is larger than that of the external drive, anticipative tracking occurs; (2) an increase in the SFA amplitude can enhance the capability of a CANN to achieve an anticipative tracking with a longer anticipative time and (3) with the proper SFA amplitudes, the network can achieve either perfect tracking or perfect anticipative tracking for a wide range of stimulus speed.

The key point for SFA achieving anticipative tracking in a CANN is that it provides a negative feedback modulation to destabilize strong localized neuronal responses. Thus, other negative feedback

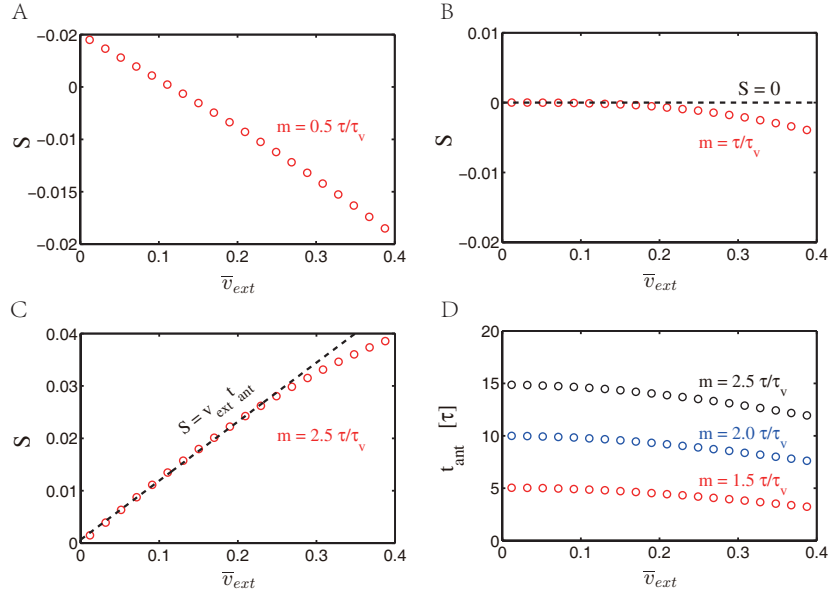

Figure 4: Tracking performances of a CANN with SFA. A. An example of delayed tracking for $m < \tau/\tau_v$; B. An example of perfect tracking for $m = \tau/\tau_v$. $s = 0$ roughly holds for a wide range of stimulus speed. C. An example of perfect anticipative tracking for $m > \tau/\tau_v$. $s$ increases linearly with $v_{\text{ext}}$ for a wide range of stimulus speed. D. Anticipative time increases with the SFA amplitude $m$. The other parameters are the same as those in Fig. 3.

modulation processes, such as short-term synaptic depression (STD) [20, 21] and negative feedback connections (NFC) from other networks [22], should also be able to realize anticipative tracking. Indeed, it was found in the previous studies that a CANN with STD or NFC can produce leading behaviors in response to moving inputs. The three mechanisms, however, have different time scales and operation levels: SFA has a time scale of one hundred milliseconds and functions at the single neuron level; STD has a time scale of hundreds to thousands of milliseconds and functions at the synapse level; and NFC has a time scale of tens of milliseconds and functions at the network level. The brain may employ them for different computational tasks in conjunction with brain functions.

It was known previously that a CANN with SFA can retain travelling wave [12]. But, to our knowledge, our study is the first one that links this intrinsic mobility of the network to the tracking performance of the neural system. We demonstrate that through regulating the SFA amplitude, a neural system can implement anticipative tracking with a range of anticipative times. Thus, it provides a flexible mechanism to compensate for a range of delay times, serving different computational purposes, e.g., by adjusting the SFA amplitudes, neural circuits along the hierarchy of a signal transmission pathway can produce increasing anticipative times, which compensate for the accumulated time delays. Our study sheds light on our understanding of how the brain processes motion information in a timely manner.

## Acknowledgments

This work is supported by grants from National Key Basic Research Program of China (NO.2014CB846101, S.W.), and National Foundation of Natural Science of China (No.11305112, Y.Y.M.; No. 31261160495, S.W.), and the Fundamental Research Funds for the central Universities (No. 31221003, S. W.), and SRFDP (No.20130003110022, S.W), and Research Grants Council of Hong Kong (Nos. 605813, 604512 and N_HKUST606/12, C.C.A.F. and K.Y.W), and Natural Science Foundation of Jiangsu Province BK20130282.

# References

[1] L. G. Nowak, M. H. J. Munk, P. Girard & J. Bullier. Visual Latencies in Areas V1 and V2 of the Macaque Monkey. *Vis. Neurosci.*, **12**, 371 – 384 (1995).

[2] C. Koch, M. Rapp & Idan Segev. A Brief History of Time (Constants). *Cereb. Cortex*, **6**, 93 – 101 (1996).

[3] H. T. Blair & P. E. Sharp. Anticipatory Head Direction Signals in Anterior Thalamus: Evidence for a Thalamocortical Circuit that Integrates Angular Head Motion to Compute Head Direction. *J. Neurosci.*, **15(9)**, 6260 – 6270 (1995).

[4] J. S. Taube & R. U. Muller. Comparisons of Head Direction Cell Activity in the Postsubiculum and Anterior Thalamus of Freely Moving Rats. *Hippocampus*, **8**, 87 – 108 (1998).

[5] J. P. Bassett, M. B. Zugaro, G. M. Muir, E. J. Golob, R. U. Muller & J. S. Taube. Passive Movements of the Head Do Not Abolish Anticipatory Firing Properties of Head Direction Cells. *J. Neurophysiol.*, **93**, 1304 – 1316 (2005).

[6] R. Nijhawan. Motion Extrapolation in Catching. *Nature*, **370**, 256 – 257 (1994).

[7] J. R. Duhamel, C. L. Colby & M. E. Goldberg. The Updating of the Representation of Visual Space in Parietal Cortex by Intended Eye Movements. *Science* **255**, 90 – 92 (1992).

[8] R. Nijhawan & S. Wu. Phil. Compensating Time Delays with Neural Predictions: Are Predictions Sensory or Motor? *Trans. R. Soc. A*, **367**, 1063 – 1078 (2009).

[9] P. E. Sharp, A. Tinkelman & Cho J. Angular Velocity and Head Direction Signals Recorded from the Dorsal Tegmental Nucleus of Gudden in the Rat: Implications for Path Integration in the Head Direction Cell Circuit. *Behav. Neurosci.*, **115**, 571 – 588 (2001).

[10] B. Gutkin & F. Zeldenrust. Spike Frequency Adaptation. *Scholarpedia*, **9**, 30643 (2014).

[11] J. Benda & A. V. M. Herz. A Universal Model for Spike-Frequency Adaptation. *Neural Comput.*, **15**, 2523 – 2564 (2003).

[12] P. C. Bressloff. Spatiotemporal Dynamics of Continuum Neural Fields. *J. Phys. A*, **45**, 033001 (2012).

[13] R. Ben-Yishai, R. L. Bar-Or & H. Sompolinsky. Theory of Orientation Tuning in Visual Cortex. *Proc. Natl. Acad. Sci. U.S.A.*, **92**, 3844 – 3848 (1995).

[14] K. Zhang. Representation of Spatial Orientation by the Intrinsic Dynamics of the Head-Direction Cell Ensemble: a Theory. *J. Neurosci.*, **16**, 2112 – 2126 (1996).

[15] A. P. Georgopoulos, M. Taira & A. Lukashin. Cognitive Neurophysiology of the Motor Cortex. *Science*, **260**, 47 – 52 (1993).

[16] A. Samsonovich & B. L. McNaughton. Path Integration and Cognitive Mapping in a Continuous Attractor Neural Network Model. *J. Neurosci*, **17**, 5900 – 5920 (1997).

[17] K. Wimmer, D. Q. Nykamp, C. Constantinidis & A. Compte. Bump Attractor Dynamics in Prefrontal Cortex Explains Behavioral Precision in Spatial Working Memory. *Nature*, **17(3)**, 431 – 439 (2014).

[18] C. C. A. Fung, K. Y. M. Wong & S. Wu. A Moving Bump in a Continuous Manifold: a Comprehensive Study of the Tracking Dynamics of Continuous Attractor Neural Networks. *Neural Comput.*, **22**, 752 – 792 (2010).

[19] J. P. Goodridge & D. S. Touretzky. Modeling attractor deformation in the rodent head direction system. *J. Neurophysio.*, **83**, 3402 – 3410 (2000).

[20] C. C. A. Fung, K. Y. M. Wong, H. Wang & S. Wu. Dynamical Synapses Enhance Neural Information Processing: Gracefulness, Accuracy, and Mobility. *Neural Comput.*, **24**, 1147 – 1185 (2012).

[21] C. C. A. Fung, K. Y. M. Wong & S. Wu. Delay Compensation with Dynamical Synapses. *Adv. in NIPS*. **25**, P. Bartlett, F. C. N. Pereira, C. J. C. Burges, L. Bottou, and K. Q. Weinberger (eds), 1097 – 1105 (2012).

[22] W. Zhang & S. Wu. Neural Information Processing with Feedback Modulations. *Neural Comput.*, **24(7)**, 1695 – 1721 (2012).

